# Enhancing Q-Learning for Optimal Asset Allocation

**Ralph Neuneier**
Siemens AG, Corporate Technology
D-81730 München, Germany
Ralph.Neuneier@mchp.siemens.de

## Abstract

This paper enhances the Q-learning algorithm for optimal asset allocation proposed in (Neuneier, 1996 [6]). The new formulation simplifies the approach by using only one value-function for many assets and allows model-free policy-iteration. After testing the new algorithm on real data, the possibility of risk management within the framework of Markov decision problems is analyzed. The proposed methods allows the construction of a multi-period portfolio management system which takes into account transaction costs, the risk preferences of the investor, and several constraints on the allocation.

## 1 Introduction

Asset allocation and portfolio management deal with the distribution of capital to various investment opportunities like stocks, bonds, foreign exchanges and others. The aim is to construct a portfolio with a maximal expected return for a given risk level and time horizon while simultaneously obeying institutional or legally required constraints. To find such an optimal portfolio the investor has to solve a difficult optimization problem consisting of two phases [4]. First, the expected yields together with a certainty measure has to be predicted. Second, based on these estimates, *mean-variance* techniques are typically applied to find an appropriate fund allocation. The problem is further complicated if the investor wants to revise her/his decision at every time step and if transaction costs for changing the allocations must be considered.

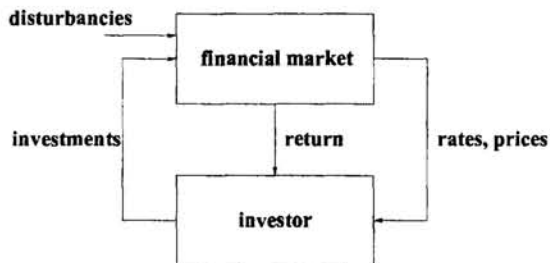

**Markov Decision Problem:**
$x_t = (\$_t, K_t)'$    state: market $\$_t$ and portfolio $K_t$
$a_t = \mu(x_t)$    policy $\mu$, actions
$p(x_{t+1}|x_t)$    transition probabilities
$r(x_t, a_t, \$_{t+1})$    return function

Within the framework of *Markov Decision Problems*, MDPs, the modeling phase and the search for an optimal portfolio can be combined (fig. above). Furthermore, transaction costs, constraints, and decision revision are naturally integrated. The theory of MDPs formalizes control problems within stochastic environments [1]. If the discrete state space is small and if an accurate model of the system is available, MDPs can be solved by con-

ventional *Dynamic Programming*, DP. On the other extreme, reinforcement learning methods using function approximator and stochastic approximation for computing the relevant expectation values can be applied to problems with large (continuous) state spaces and without an appropriate model available [2, 10].

In [6], asset allocation is formalized as a MDP under the following assumptions which clarify the relationship between MDP and portfolio optimization:

1. The investor may trade at each time step for an infinite time horizon.
2. The investor is not able to influence the market by her/his trading.
3. There are only two possible assets for investing the capital.
4. The investor has no risk aversion and always invests the total amount.

The reinforcement algorithm *Q-Learning*, QL, has been tested on the task to invest liquid capital in the German stock market DAX, using neural networks as value function approximators for the Q-values $Q(x, a)$. The resulting allocation strategy generated more profit than a heuristic benchmark policy [6].

Here, a new formulation of the QL algorithm is proposed which allows to relax the third assumption. Furthermore, in section 3 the possibility of risk control within the MDP framework is analyzed which relaxes assumption four.

## 2 Q-Learning with uncontrollable state elements

This section explains how the QL algorithm can be simplified by the introduction of an artificial deterministic transition step. Using real data, the successful application of the new algorithm is demonstrated.

### 2.1 Q-Learning for asset allocation

The situation of an investor is formalized at time step $t$ by the state vector $x_t = (\$_t, K_t)$, which consists of elements $\$_t$ describing the financial market (e. g. interest rates, stock indices), and of elements $K_t$ describing the investor's current allocation of the capital (e. g. how much capital is invested in which asset). The investor's decision $a_t$ for a new allocation and the dynamics on the financial market let the state switch to $x_{t+1} = (\$_{t+1}, K_{t+1})$ according to the transition probability $p(x_{t+1}|x_t, a_t)$. Each transition results in an immediate return $r_t = r(x_t, x_{t+1}, a_t)$ which incorporates possible transaction costs depending on the decision $a_t$ and the change of the value of $K_t$ due to the new values of the assets at time $t + 1$. The aim is to maximize the expected discounted sum of the returns, $V^*(x) = E(\sum_{t=0}^{\infty} \gamma^t r_t | x_0 = x)$, by following an optimal stationary policy $\mu^*(x_t) = a_t$. For a discrete finite state space the solution can be stated as the recursive Bellman equation:

$$V^*(x_t) = \max_a \left[ \sum_{x_{t+1}} p(x_{t+1}|x_t, a)r_t + \gamma \sum_{x_{t+1}} p(x_{t+1}|x_t, a)V^*(x_{t+1}) \right]. \quad (1)$$

A more useful formulation defines a Q-function $Q^*(x, a)$ of state-action pairs $(x_t, a_t)$,

$$Q^*(x_t, a_t) := \sum_{x_{t+1}} p(x_{t+1}|x_t, a_t)r_t + \gamma \sum_{x_{t+1}} p(x_{t+1}|x_t, a_t) \max_{a \in A}(Q^*(x_{t+1}, a)), \quad (2)$$

to allow the application of an iterative stochastic approximation scheme, called Q-Learning [11]. The Q-value $Q^*(x_t, a_t)$ quantifies the expected discounted sum of returns if one executes action $a_t$ in state $x_t$ and follows an optimal policy thereafter, i. e. $V^*(x_t) = \max_a Q^*(x_t, a)$. Observing the tuple $(x_t, x_{t+1}, a_t, r_t)$, the tabulated Q-values are updated

in the $k + 1$ iteration step with learning rate $\eta_k$ according to:

$$\text{QL: } Q_{k+1}(x_t, a_t) = (1 - \eta_k)Q_k(x_t, a_t) + \eta_k(r_t + \gamma \max_{a \in A}(Q_k(x_{t+1}, a))) \,.$$

It can be shown, that the sequence of $Q_k$ converges under certain assumptions to $Q^*$. If the Q-values $Q^*(x, a)$ are approximated by separate neural networks with weight vector $w^a$ for different actions $a$, $Q^*(x, a) \approx Q(x; w^a)$, the adaptations (called NN-QL) are based on the temporal differences $d_t$:

$$d_t := r(x_t, a_t, x_{t+1}) + \gamma \max_{a \in A} Q(x_{t+1}; w_k^a) - Q(x_t; w_k^{a_t}) \,,$$

$$\text{NN-QL: } w_{k+1}^{a_t} = w_k^{a_t} + \eta_k d_t \nabla Q(x_t; w_k^{a_t}) \,.$$

Note, that although the market dependent part $\$_t$ of the state vector is independent of the investor's decisions, the future wealth $K_{t+1}$ and the returns $r_t$ are not. Therefore, asset allocation is a multi-stage decision problem and may not be reduced to pure prediction if transaction costs must be considered. On the other hand, the attractive feature that the decisions do not influence the market allows to approximate the Q-values using historical data of the financial market. We need not to invest real money during the training phase.

## 2.2 Introduction of an artificial deterministic transition

Now, the Q-values are reformulated in order to make them independent of the actions chosen at the time step $t$. Due to assumption 2, which states that the investor can not influence the market by the trading decisions, the stochastic process of the dynamics of $\$_t$ is an uncontrollable Markov chain. This allows the introduction of a deterministic intermediate step between the transition from $x_t$ to $x_{t+1}$ (see fig. below). After the investor has chosen an action $a_t$, the capital $K_t$ changes to $K'_t$ because he/she may have paid transaction costs $c_t = c(K_t, a_t)$ and $K'_t$ reflects the new allocation whereas the state of the market, $\$_t$, remains the same. Because the costs $c_t$ are known in advance, this transition is deterministic and controllable. Then, the market switches stochastically to $\$_{t+1}$ and generates the immediate return $r'_t = r'(\$_t, K'_t, \$_{t+1})$ i.e., $r_t = c_t + r'_t$. The capital changes to $K_{t+1} = r'_t + K'_t$. This transition is uncontrollable by the investor. $V^*(\$, K) = V^*(x)$ is now computed using the costs $c_t$ and returns $r'_t$ (compare also eq. 1)

$$V^*(\$, K) = \max_{a_0, \dots} E \left[ \sum_{t=0}^{\infty} \gamma^t (c(K_t, a_t) + r'(\$_t, K'_t, \$_{t+1})) \, \middle| \, \begin{matrix} \$_0 = \$ \\ K_0 = K \end{matrix} \right] \,.$$

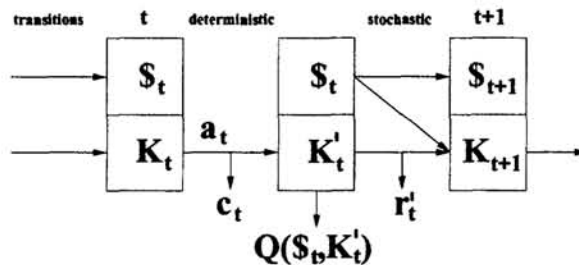

Defining $Q^*(\$_t, K'_t)$ as the Q-values of the intermediate time step

$$
\begin{aligned}
Q^*(\$_t, K'_t) &:= E\left[r'(\$_t, K'_t, \$_{t+1}) + \gamma V^*(\$_{t+1}, K_{t+1})\right] \,, \\
&= E[r'_t + \gamma \max_{a_{t+1}}[c_{t+1} + Q^*(\$_{t+1}, K'_{t+1})]] \,,
\end{aligned}
$$

gives rise to the optimal value function and policy (time indices are suppressed),

$$V^*(\$, K) = \max_a[c(K, a) + Q^*(\$, K')],$$

$$\mu^*(\$, K) = \arg\max_a[c(K, a) + Q^*(\$, K')].$$

Defining the temporal differences $d_t$ for the approximation $Q_k$ as

$$d_t := r'(\$_t, K'_t, \$_{t+1}) + \gamma \max_a[c(K_{t+1}, a) + Q^{(k)}(\$_{t+1}, K'_{t+1})] - Q^{(k)}(\$_t, K'_t)$$

leads to the update equations for the Q-values represented by tables or networks:

QLU: $\quad Q^{(k+1)}(\$_t, K'_t) = Q^{(k)}(\$_t, K'_t) + \eta_k d_t,$

NN-QLU: $\quad w^{(k+1)} = w^{(k)} + \eta_k d_t \nabla Q(\$, K'; w^{(k)}).$

The simplification is now obvious, because (NN-)QLU only needs one table or neural network no matter how many assets are concerned. This may lead to a faster convergence and better results. The training algorithm boils down to the iteration of the following steps:

---

**QLU for optimal investment decisions**

1. draw randomly patterns $\$_t$, $\$_{t+1}$ from the data set, draw randomly an asset allocation $K'_t$

2. for all possible actions $a$: compute $r'_t, c(K_{t+1}, a), Q^{(k)}(\$_{t+1}, K'_{t+1})$

3. compute temporal difference $d_t$

4. compute new value $Q^{(k+1)}(\$_t, K'_t)$ resp. $Q(\$_t, K'_t; w^{(k+1)})$

5. stop, if Q-values have converged, otherwise go to 1

---

Since QLU is equivalent to Q-Learning, QLU converges to the optimal Q-values under the same conditions as QL (e. g [2]). The main advantage of (NN-)QLU is that this algorithm only needs one value function no matter how many assets are concerned and how fine the grid of actions are:

$$Q^*((\$, K), a) = c(K, a) + Q^*(\$, K').$$

Interestingly, the convergence to an optimal policy of QLU does not rely on an explicit exploration strategy because the randomly chosen capital $K'_t$ in step 1 simulates a random action which was responsible for the transition from $K_t$. In combination with the randomly chosen market state $\$_t$, a sufficient exploration of the action and state space is guaranteed.

### 2.3 Model-free policy-iteration
The reformulation also allows the design of a policy iteration algorithm by alternating a policy evaluation phase (PE) and a policy improvement (PI) step. Defining the temporal differences $d_t$ for the approximation $Q_k^{\mu_l}$ of the policy $\mu_l$ in the k step of PE

$$d_t := r'(\$_t, K'_t, \$_{t+1}) + \gamma[c(K_{t+1}, \mu_l(\$_{t+1}, K_{t+1})) + Q^{(k)}(K'_{t+1}, \$_{t+1})] - Q^{(k)}(K'_t, \$_t)$$

leads to the following update equation for tabulated Q-values

$$Q_{\mu_l}^{(k+1)}(\$_t, K'_t) = Q_{\mu_l}^{(k)}(\$_t, K'_t) + \eta_k d_t.$$

After convergence, one can improve the policy $\mu_l$ to $\mu_{l+1}$ by

$$\mu_{l+1}(\$_t, K_t) = \arg\max_a [c(K_t, a) + Q^{\mu_l}(\$_t, K'_t)] .$$

By alternating the two steps PE and PI, the sequence of policies $[\mu_l(x)]_{l=0,...}$ converges under the typical assumptions to the optimal policy $\mu^*(x)$ [2].

Note, that policy iteration is normally not possible using classical QL, if one has not an appropriate model at hand. The introduction of the deterministic intermediate step allows to start with an initial strategy (e. g. given by a broker), which can be subsequently optimized by model-free policy iteration trained with historical data of the financial market. Generalization to parameterized value functions is straightforward.

### 2.4 Experiments on the German Stock Index DAX

The NN-QLU algorithm is now tested on a real world task: assume that an investor wishes to invest her/his capital into a portfolio of stocks which behaves like the German stock index DAX. Her/his alternative is to keep the capital in the certain asset cash, referred to as DM. We compare the resulting strategy with three benchmarks, namely Neuro-Fuzzy, Buy&Hold and the naive prediction. The Buy&Hold strategy invests at the first time step in the DAX and only sells at the end. The naive prediction invests if the past return of the DAX has been positive and v. v. The third is based on a Neuro-Fuzzy model which was optimized to predict the daily changes of the DAX [8]. The heuristic benchmark strategy is then constructed by taking the sign of the prediction as a trading signal, such that a positive prediction leads to an investment in stocks. The input vector of the Neuro-Fuzzy model, which consists of the DAX itself and 11 other influencing market variables, was carefully optimized for optimal prediction. These inputs also constitutes the $\$_t$ part of the state vector which describes the market within the NN-QLU algorithm. The data is split into a training (from 2. Jan. 1986 to 31. Dec. 1994) and a test set (from 2. Jan. 1993 to 1. Aug. 1996). The transaction costs $(c_t)$ are 0.2% of the invested capital if $K_t$ is changed from DM to DAX, which are realistic for financial institutions. Referring to an epoch as one loop over all training patterns, the training proceeds as outlined in the previous section for 10000 epochs with $\eta_k = \eta_0 \cdot 0.999^k$ with start value $\eta_0 = 0.05$.

Table 1: Comparison of the profitability of the strategies, the number of position changes and investments in DAX for the test (training) data.

| strategy | profit | investments in DAX | position changes |
|---|---|---|---|
| NN-QLU | 1.60 (3.74) | 70 (73)% | 30 (29)% |
| Neuro-Fuzzy | 1.35 (1.98) | 53 (53)% | 50 (52)% |
| Naive Prediction | 0.80 (1.06) | 51 (51)% | 51 (48)% |
| Buy&Hold | 1.21 (1.46) | 100 (100)% | 0 (0)% |

The strategy constructed with the NN-QLU algorithm, using a neural network with 8 hidden neurons and a linear output, clearly beats the benchmarks. The capital at the end of the test set (training set) exceeds the second best strategy Neuro-Fuzzy by about 18.5% (89%) (fig. 1). One reason for this success is, that QLU changes less often the position and thus, avoids expensive transaction costs. The Neuro-Fuzzy policy changes almost every second day whereas NN-QLU changes only every third day (see tab. 1).

It is interesting to analyze the learning behavior during training by evaluating the strategies of NN-QLU after each epoch. At the beginning, the policies suggest to change almost never or each time to invest in DAX. After some thousand epochs, these bang-bang strategies starts to differentiate. Simultaneously, the more complex the strategies become the more profit they generate (fig. 2).

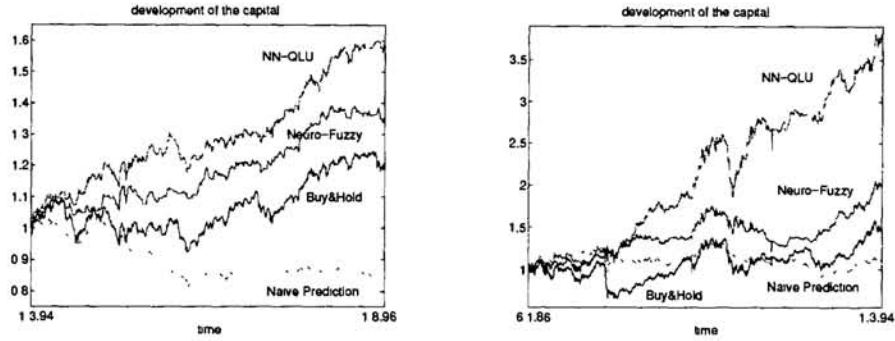

Figure 1: Comparison of the development of the capital for the test set (left) and the training set (right). The NN-QLU strategy clearly beats all the benchmarks.

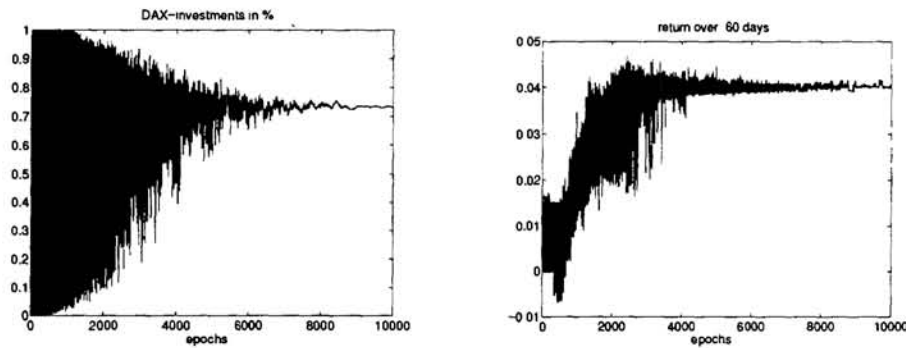

Figure 2: Training course: percentage of DAX investments (left), profitability measured as the average return over 60 days on the training set (right).

## 3 Controlling the Variance of the Investment Strategies

### 3.1 Risk-adjusted MDPs

People are not only interested in maximizing the return, but also in controlling the risk of their investments. This has been formalized in the Markowitz portfolio-selection, which aims for an allocation with the maximal expected return for a given risk level [4]. Given a stationary policy $\mu(x)$ with finite state space, the associated value function $V^\mu(x)$ and its variance $\sigma^2(V^\mu(x))$ can be defined as

$$V^\mu(x) \;=\; E\left[\left.\sum_{t=0}^{\infty}\gamma^t r(x_t,\mu_t,x_{t+1})\right|x_0=x\right],$$

$$\sigma^2(V^\mu(x)) \;=\; E\left[\left.\left(\sum_{t=0}^{\infty}\gamma^t r(x_t,\mu_t,x_{t+1})-V^\mu(x)\right)^2\right|x_0=x\right].$$

Then, an optimal strategy $\mu^*(x;\lambda)$ for a *risk-adjusted MDP* (see [9], S. 410 for variance-penalized MDPs) is

$$\mu^*(x;\lambda) = \arg\max_\mu[V^\mu(x)-\lambda\,\sigma^2(V^\mu(x))] \quad \text{for } \lambda \geq 0\,.$$

By variation of $\lambda$, one can construct so-called efficient portfolios which have minimal risk for each achievable level of expected return. But in comparison to classical portfolio theory, this approach manages multi-period portfolio management systems including transaction costs. Furthermore, typical min-max requirements on the trading volume and other allocation constraints can be easily implemented by constraining the action space.

### 3.2   Non-linear Utility Functions

In general, it is not possible to compute $\sigma^2(V^\mu(x))$ with (approximate) dynamic programming or reinforcement techniques, because $\sigma^2(V^\mu(x))$ can not be written in a recursive Bellman equation. One solution to this problem is the use of a return function $r_t$, which penalizes high variance. In financial analysis, the Sharpe-ratio, which relates the mean of the single returns to their variance i. e., $\bar{r}/\sigma(r)$, is often employed to describe the smoothness of an equity curve. For example, Moody has developed a Sharpe-ratio based error function and combines it with a recursive training procedure [5] (see also [3]). The limitation of the Sharpe-ratio is, that it penalizes also upside volatility. For this reason, the use of an utility function with a negative second derivative, typical for risk averse investors, seems to be more promising. For such return functions an additional unit increase is less valuable than the last unit increase [4]. An example is $r = log$(new portfolio value / old portfolio value) which also penalizes losses much stronger than gains. The Q-function $Q(x, a)$ may lead to intermediate values of $a^*$ as shown in the figure below.

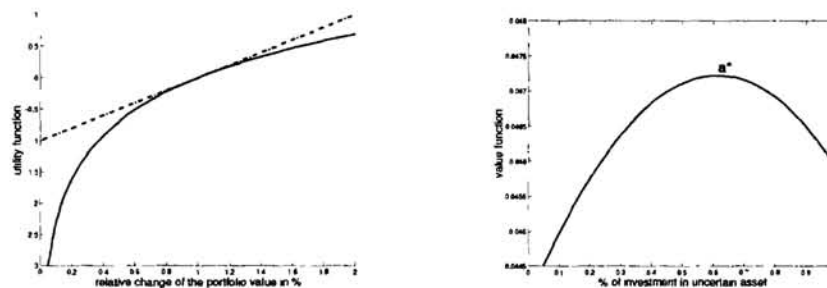

### 4   Conclusion and Future Work

Two improvements of Q-learning have been proposed to bridge the gap between classical portfolio management and asset allocation with adaptive dynamic programming. It is planned to apply these techniques within the framework of a European Community sponsored research project in order to design a decision support system for strategic asset allocation [7]. Future work includes approximations and variational methods to compute explicitly the risk $\sigma^2(V^\mu(x))$ of a policy.

### References

[1]   D. P. Bertsekas. *Dynamic Programming and Optimal Control*, vol. 1. Athena Scientific, 1995.

[2]   D. P. Bertsekas and J. N. Tsitsiklis. *Neuro-Dynamic Programming*. Athena Scientific, 1996.

[3]   M. Choey and A. S. Weigend. Nonlinear trading models through Sharpe Ratio maximization. In proc. of NNCM'96, 1997. World Scientific.

[4]   E. J. Elton and M. J. Gruber. *Modern Portfolio Theory and Investment Analysis*. 1995.

[5]   J. Moody, L. Whu, Y. Liao, and M. Saffell. Performance Functions and Reinforcement Learning for Trading Systems and Portfolios. *Journal of Forecasting*, 1998. forthcoming.

[6]   R. Neuneier. Optimal asset allocation using adaptive dynamic programming. In proc. of *Advances in Neural Information Processing Systems*, vol. 8, 1996.

[7]   R. Neuneier, H. G. Zimmermann, P. Hierve, and P. Naim. Advanced Adaptive Asset Allocation. *EU Neuro-Demonstrator*, 1997.

[8]   R. Neuneier, H. G. Zimmermann, and S. Siekmann. Advanced Neuro-Fuzzy in Finance: Predicting the German Stock Index DAX, 1996. Invited presentation at ICONIP'96, Hong Kong, available by email from Ralph.Neuneier@mchp.siemens.de.

[9]   M. L. Puterman. *Markov Decision Processes*. John Wiley & Sons, 1994.

[10]   S. P. Singh. *Learning to Solve Markovian Decision Processes*. CMPSCI TR 93-77, University of Massachusetts, November 1993.

[11]   C. J. C. H. Watkins and P. Dayan. Technical Note: Q-Learning. *Machine Learning: Special Issue on Reinforcement Learning*, 8, 3/4:279–292, May 1992.
